# Function Approximation with the Sweeping Hinge Algorithm

**Don R. Hush, Fernando Lozano**
Dept. of Elec. and Comp. Engg.
University of New Mexico
Albuquerque, NM 87131

**Bill Horne**
MakeWaves, Inc.
832 Valley Road
Watchung, NJ 07060

## Abstract

We present a computationally efficient algorithm for function approximation with piecewise linear sigmoidal nodes. A one hidden layer network is constructed one node at a time using the method of fitting the residual. The task of fitting individual nodes is accomplished using a new algorithm that searchs for the best fit by solving a sequence of Quadratic Programming problems. This approach offers significant advantages over derivative–based search algorithms (e.g. backpropagation and its extensions). Unique characteristics of this algorithm include: finite step convergence, a simple stopping criterion, a deterministic methodology for seeking "good" local minima, good scaling properties and a robust numerical implementation.

## 1 Introduction

The learning algorithm developed in this paper is quite different from the traditional family of derivative–based descent methods used to train Multilayer Perceptrons (MLPs) for function approximation. First, a *constructive* approach is used, which builds the network one node at a time. Second, and more importantly, we use *piecewise linear* sigmoidal nodes instead of the more popular (continuously differentiable) logistic nodes. These two differences change the nature of the learning problem entirely. It becomes a *combinatorial* problem in the sense that the number of feasible solutions that must be considered in the search is *finite*. We show that this number is exponential in the input dimension, and that the problem of finding the global optimum admits no polynomial–time solution. We then proceed to develop a heuristic algorithm that produces good approximations with reasonable efficiency. This algorithm has a simple stopping criterion, and very few user specified parameters. In addition, it produces solutions that are comparable to (and sometimes better than) those produced by local descent methods, and it does so

using a deterministic methodology, so that the results are independent of initial conditions.

## 2   Background and Motivation

We wish to approximate an unknown continuous function $f(\mathbf{x})$ over a compact set with a one–hidden layer network described by

$$\hat{f}_n(\mathbf{x}) = a_0 + \sum_{i=1}^{n} a_i \sigma(\mathbf{x}, \mathbf{w}_i) \tag{1}$$

where $n$ is the number of hidden layer nodes (basis functions), $\mathbf{x} \in \Re^d$ is the input vector, and $\{\sigma(\mathbf{x}, \mathbf{w})\}$ are sigmoidal functions parameterized by a weight vector $\mathbf{w}$. A set of example data, $S = \{\mathbf{x}_i, y_i\}$, with a total of $N$ samples is available for training and test.

The models in (1) have been shown to be universal approximators. More importantly, (Barron, 1993) has shown that for a special class of continuous functions, $\Gamma_C$, the generalization error satisfies

$$E[\|f - f_{n,N}\|^2] \ \le \|f - f_n\|^2 + E[\|f_n - f_{n,N}\|^2]$$

$$= O\left(\tfrac{1}{n}\right) + O\left(\tfrac{nd \log N}{N}\right)$$

where $\|\cdot\|$ is the appropriate two–norm, $f_n$ is the the best $n$–node approximation to $f$, and $f_{n,N}$ is the approximation that best fits the samples in $S$. In this equation $\|f - f_n\|^2$ and $E[\|f_n - f_{n,N}\|^2]$ correspond to the *approximation* and *estimation* error respectively. Of particular interest is the $O(1/n)$ bound on approximation error, which for fixed basis functions is of the form $O(1/n^{2/d})$ (Barron, 1993). Barron's result tells us that the (tunable) sigmoidal bases are able to avoid the curse of dimensionality (for functions in $\Gamma_C$). Further, it has been shown that the $O(1/n)$ bound can be achieved *constructively* (Jones, 1992), that is by designing the basis functions (nodes) one at a time. The proof of this result is itself constructive, and thus provides a framework for the development of an algorithm which can (in principle) achieve this bound. One manifestation of this algorithm is shown in Figure 1. We call this the *iterative approximation algorithm* (IIA) because it builds the approximation by iterating on the residual (i.e. the unexplained portion of the function) at each step. This is the same algorithmic strategy used to form bases in numerous other settings, e.g. Grahm-Schmidt, Conjugate Gradient, and Projection Pursuit. The difficult part of the IIA algorithm is in the determination of the best fitting basis function $\sigma_n$ in step 2. This is the focus of the remainder of this paper.

## 3   Algorithmic Development

We begin by defining the *hinging sigmoid* (HS) node on which our algorithms are based. An HS node performs the function

$$\sigma_h(\mathbf{x}, \mathbf{w}) = \begin{cases} w_+, & \tilde{\mathbf{w}}_l^T \tilde{\mathbf{x}} \ge w_+ \\ \tilde{\mathbf{w}}_l^T \tilde{\mathbf{x}}, & w_- \le \tilde{\mathbf{w}}_l^T \tilde{\mathbf{x}} \le w_+ \\ w_-, & \tilde{\mathbf{w}}_l^T \tilde{\mathbf{x}} \le w_- \end{cases} \tag{2}$$

where $\mathbf{w}^T = [\tilde{\mathbf{w}}_l \ w_+ \ w_-]$ and $\tilde{\mathbf{x}}$ is an augmented input vector with a 1 in the first component. An example of the surface formed by an HS node on a two–dimensional input is shown in Figure 2. It is comprised of three hyperplanes joined pairwise

**Initialization:** $f_0(\mathbf{x}) = 0$
**for** $n = 1$ **to** $n_{max}$ **do**
     1. Compute Residual:     $e_n(\mathbf{x}) = f(\mathbf{x}) - f_{n-1}(\mathbf{x})$
     2. Fit Residual:         $\sigma_n(\mathbf{x}) = \arg\min_{\sigma \in \Sigma} \|e_n(\mathbf{x}) - \sigma(\mathbf{x})\|$
     3. Update Estimate:    $f_n(\mathbf{x}) = \alpha f_{n-1}(\mathbf{x}) + \beta \sigma_n(\mathbf{x})$
        where $\alpha$ and $\beta$ are chosen to minimize $\|f(\mathbf{x}) - f_n(\mathbf{x})\|$
**endloop**

Figure 1: Iterative Approximation Algorithm (IIA).

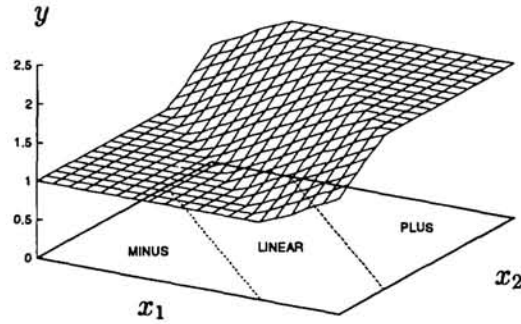

Figure 2: A Sigmoid Hinge function in two dimensions.

continuously at two hinge locations. The upper and middle hyperplanes are joined at "Hinge 1" and the lower and middle hyperplanes are joined at "Hinge 2". These hinges induce linear partitions on the input space that divide the space into three regions, and the samples in $S$ into three subsets,

$$
\begin{aligned}
S_+ &= \left\{ (\mathbf{x}_i, y_i) : \tilde{\mathbf{w}}_l^T \tilde{\mathbf{x}}_i \geq w_+ \right\} \\
S_l &= \left\{ (\mathbf{x}_i, y_i) : w_- \leq \tilde{\mathbf{w}}_l^T \tilde{\mathbf{x}}_i \leq w_+ \right\} \\
S_- &= \left\{ (\mathbf{x}_i, y_i) : \tilde{\mathbf{w}}_l^T \tilde{\mathbf{x}}_i \leq w_- \right\}
\end{aligned}
\tag{3}
$$

These subsets, and the corresponding regions of the input space, are referred to as the PLUS, LINEAR and MINUS subsets/regions respectively. We refer to this type of partition as a *sigmoidal partition*. A sigmoidal partition of $S$ will be denoted $P = \{S_+, S_l, S_-\}$, and the set of all such partitions will be denoted $\Pi = \{P_i\}$. Input samples that fall on the boundary between two regions can be assigned to the set on either side. These points are referred to as *hinge samples* and play a crucial role in subsequent development. Note that once a weight vector $\mathbf{w}$ is specified, the partition $P$ is completely determined, but the reverse is not necessarily true. That is, there are generally an infinite number of weight vectors that induce the same partition.

We begin our quest for a learning algorithm with the development of an expression for the empirical risk. The empirical risk (squared error over the sample set) is defined

$$
E_P(\mathbf{w}) = \frac{1}{2} \sum_S (y_i - \sigma_h(\mathbf{x}_i, \mathbf{w}))^2
\tag{4}
$$

This expression can be expanded into three terms, one for each set in the partition,

$$E_P(\mathbf{w}) = \frac{1}{2}\sum_{S_-}(y_i - w_-)^2 + \frac{1}{2}\sum_{S_+}(y_i - w_+)^2 + \frac{1}{2}\sum_{S_l}(y_i - \tilde{\mathbf{w}}_l^T \tilde{\mathbf{x}}_i)^2$$

After further expansion and rearrangement of terms we obtain

$$E_P(\mathbf{w}) = \frac{1}{2}\mathbf{w}^T\mathbf{R}\mathbf{w} - \mathbf{w}^T\mathbf{r} + s_y^2 \qquad (5)$$

where

$$\mathbf{R}_l = \sum_{S_l}\tilde{\mathbf{x}}_i\tilde{\mathbf{x}}_i^T \quad \mathbf{r}_l = \sum_{S_l}\tilde{\mathbf{x}}_i y_i \qquad (6)$$

$$s_y^2 = \frac{1}{2}\sum_S y_i^2 \quad s_y^+ = \sum_{S_+} y_i \quad s_y^- = \sum_{S_-} y_i \qquad (7)$$

$$\mathbf{R} = \begin{pmatrix} \mathbf{R}_l & 0 & 0 \\ 0 & N_+ & 0 \\ 0 & 0 & N_- \end{pmatrix} \quad \mathbf{r} = \begin{pmatrix} \mathbf{r}_l \\ s_y^+ \\ s_y^- \end{pmatrix} \quad \mathbf{w} = \begin{pmatrix} \tilde{\mathbf{w}}_l \\ w_+ \\ w_- \end{pmatrix} \qquad (8)$$

and $N_+$, $N_l$ and $N_-$ are the number of samples in $S_+$, $S_l$ and $S_-$ respectively. The subscript $P$ is used to emphasize that this criterion is dependent on the partition (i.e. $P$ is required to form $\mathbf{R}$ and $\mathbf{r}$). In fact, the nature of the partition plays a critical role in determining the properties of the solution. When $\mathbf{R}$ is positive *definite* (i.e. full rank), $P$ is referred to as a *stable* partition, and when $\mathbf{R}$ has reduced rank $P$ is referred to as an *unstable* partition. A stable partition requires that $\mathbf{R}_l > 0$. For purposes of algorithm development we will assume that $\mathbf{R}_l > 0$ when $|S_l| > N_{min}$, where $N_{min}$ is a suitably chosen value greater than or equal to $d + 1$. With this, a necessary condition for a stable partition is that there be at least one sample in $S_+$ and $S_-$ and $N_l \geq N_{min}$. When seeking a minimizing solution for $E_P(\mathbf{w})$ we restrict ourselves to stable partitions because of the potential nonuniqueness associated with solutions to unstable partitions.

Determining a weight vector that simultaneously minimizes $E_P(\mathbf{w})$ and preserves the current partition can be posed as a constrained optimization problem. This problem takes on the form

$$\min \frac{1}{2}\mathbf{w}^T\mathbf{R}\mathbf{w} - \mathbf{w}^T\mathbf{r}$$
$$\text{subject to } \mathbf{A}\mathbf{w} \leq \mathbf{0} \qquad (9)$$

where the inequality constraints are designed to maintain the current partition defined by (3). This is a *Quadratic Programming* problem with *inequality constraints*, and because $\mathbf{R} > 0$ it has a unique global minimum. The general Quadratic Programming problem is $NP$–hard and also hard to approximate (Bellare and Rogaway, 1993). However, the convex case which we restrict ourselves to here (i.e. $\mathbf{R} > 0$) admits a polynomial time solution. In this paper we use the *active set* algorithm (Luenberger, 1984) to solve (9). With the proper implementation, this algorithm runs in $O(k(d^2 + Nd))$ time, where $k$ is typically on the order of $d$ or less.

The solution to the quadratic programming problem in (9) is only as good as the current partition allows. The more challenging aspect of minimizing $E_P(\mathbf{w})$ is in the search for a good partition. Unfortunately there is no ordering or arrangement of partitions that is convex in $E_P(\mathbf{w})$, so the search for the optimal partition will be a computationally challenging problem. An exhaustive search is usually out of the question because of the prohibitively large number of partitions, as given by the following lemma.

**Lemma 1:** *Let $S$ contain a total of $N$ samples in $\Re^d$ that lie in general position. Then the number of sigmoidal partitions defined in (3) is $\Theta(N^{d+1})$.*

**Proof:** A detailed proof is beyond the scope of this paper, but an intuitive proof follows. It is well–known that the number of linear dichotomies of $N$ points in $d$ dimensions is $\Theta(N^d)$ (Edelsbrunner, 1987). Each sigmoidal partition is comprised of two linear dichotomies, one formed by Hinge 1 and the other by Hinge 2, and these dichotomies are constrained to be simple translations of one another. Thus, to enumerate all sigmoidal partitions we allow one of the hinges, say Hinge 1, can take on $\Theta(N^d)$ different positions. For each of these the other hinge can occupy only $\sim N$ unique positions. The total is therefore $\Theta(N^{d+1})$.

The search algorithm developed here employs a Quadratic Programming (QP) algorithm at each new partition to determine the optimal weight vector for that partition (i.e. the optimal orientation for the separating hyperplanes). Transitions are made from one partition to the next by allowing hinge samples to flip from one side of the hinge boundary to the next. The search is terminated when a minimum value of $E_P(\mathbf{w})$ is found (i.e. it can no longer be reduced by flipping hinge samples). Such an algorithm is shown in Figure 3. We call this the `HingeDescent` algorithm because it allows the hinges to "walk across" the data in a manner that descends the $E_P(\mathbf{w})$ criterion. Note that provisions are made within the algorithm to avoid unstable partitions. Note also that it is easy to modify this algorithm to descend only one hinge at a time, simply by omitting one of the blocks of code that flips samples across the corresponding hinge boundary.

```
{This routine is invoked with a stable feasible solution W = {w, R, r, A, S₊, Sₗ, S₋}.}
procedure HingeDescent (W)
    { Allow hinges to walk across the data until a minimizing partition is found. }
    E = ½wᵀRw − wᵀr
    do
        Eₘᵢₙ = E
        {Flip Hinge 1 Samples.}
        for each ((xᵢ, yᵢ) on Hinge 1) do
            if ((xᵢ, yᵢ) ∈ S₊ and N₊ > 1) then
                Move (xᵢ, yᵢ) from S₊ to Sₗ, and update R, r, and A
            elseif ((xᵢ, yᵢ) ∈ Sₗ and Nₗ > Nₘᵢₙ) then
                Move (xᵢ, yᵢ) from Sₗ to S₊, and update R, r, and A
            endif
        endloop
        {Flip Hinge 2 Samples.}
        for each ((xᵢ, yᵢ) on Hinge 2) do
            if ((xᵢ, yᵢ) ∈ S₋ and N₋ > 1) then
                Move (xᵢ, yᵢ) from S₋ to Sₗ, and update R, r, and A
            elseif ((xᵢ, yᵢ) ∈ Sₗ and Nₗ > Nₘᵢₙ) then
                Move (xᵢ, yᵢ) from Sₗ to S₋, and update R, r, and A
            endif
        endloop
        {Compute optimal solution for new partition.}
        W = QPSolve(W);
        E = ½wᵀRw − wᵀr
    while (E < Eₘᵢₙ) ;
    return(W);
end ;    {HingeDescent}
```

Figure 3: The `HingeDescent` Algorithm.

**Lemma 2:** *When started at a stable partition, the* `HingeDescent` *algorithm will*

*converge to a stable partition of $E_P(\mathbf{w})$ in a finite number of steps.*

**Proof:** First note that when $\mathbf{R} > 0$, a QP solution can always be found in a finite number of steps. The proof of this result is beyond the scope of this paper, but can easily be found in the literature (Luenberger, 1984). Now, by design, `HingeDescent` always moves from one stable partition to the next, maintaining the $\mathbf{R} > 0$ property at each step so that all QP solutions can be produced in a finite number of steps. In addition, $E_P(\mathbf{w})$ is reduced at each step (except the last one) so no partitions are revisited, and since there are a finite number of partitions (see Lemma 1) this algorithm must terminate in a finite number of steps. QED.

Assume that `QPSolve` runs in $O(k(d^2+Nd))$ time as previously stated. Then the run time of `HingeDescent` is given by $O(N_p((k+N_h)d^2+kNd))$, where $N_h$ is the number of samples flipped at each step and $N_p$ is the total number of partitions explored. Typical values for $k$ and $N_h$ are on the order of $d$, simplifying this expression to $O(N_p(d^3 + Nd^2))$. $N_p$ can vary widely, but is often substantially less than $N$.

`HingeDescent` seeks a local minimum over $\Pi$, and may produce a poor solution, depending on the starting partition. One way to remedy this is to start from several different initial partitions, and then retain the best solution overall. We take a different approach here, that always starts with the same initial condition, visits several local minima along the way, and always ends up with the same final solution each time.

The `SweepingHinge` algorithm works as follows. It starts by placing one of the hinges, say Hinge 1, at the outer boundary of the data. It then sweeps this hinge across the data, $M$ samples at a time (e.g. $M = 1$), allowing the other hinge (Hinge 2) to descend to an optimal position at each step. The initial hinge locations are determined as follows. A linear fit is formed to the entire data set and the hinges are positioned at opposite ends of the data so that the PLUS and MINUS regions meet the LINEAR region at the two data samples on either end. After the initial linear fit, the hinges are allowed to descend to a local minimum using `HingeDescent`. Then Hinge 1 is swept across the data $M$ samples at a time. Mechanically this is achieved by moving $M$ additional samples from $S_l$ to $S_+$ at each step. Hinge 2 is allowed to descend to an optimal position at each of these steps using the `Hinge2Descent` algorithm. This algorithm is identical to `HingeDescent` except that the code that flips samples across Hinge 1 is omitted. The best overall solution from the sweep is retained and "fine–tuned" with one final pass through the `HingeDescent` algorithm to produce the final solution.

The run time of `SweepingHinge` is no worse than $N/M$ times that of `HingeDescent`. Given this, an upper bound on the (typical) run time for this algorithm (with $M = 1$) is $O(NN_p(d^3 + Nd^2))$. Consequently, `SweepingHinge` scales reasonably well in both $N$ and $d$, considering the nature of the problem it is designed to solve.

## 4  Empirical Results

The following experiment was adapted from (Breiman, 1993). The function $f(\mathbf{x}) = e^{-\|\mathbf{x}\|^2}$ is sampled at $100d$ points $\{\mathbf{x}_i\}$ such that $\|\mathbf{x}\| \le 3$ and $\|\mathbf{x}\|$ is uniform on $[0, 3]$. The dimension $d$ is varied from 4 to 10 (in steps of 2) and models of size 1 to 20 nodes are trained using the IIA/`SweepingHinge` algorithm. The number of samples traversed at each step of the sweep in `SweepingHinge` was set to $M = 10$. $N_{min}$ was set equal to $3d$ throughout. A refitting pass was employed after each new node was added in the IIA. The refitting algorithm used `HingeDescent` to "fine–tune" each node each node before adding the next node. The average sum of squared

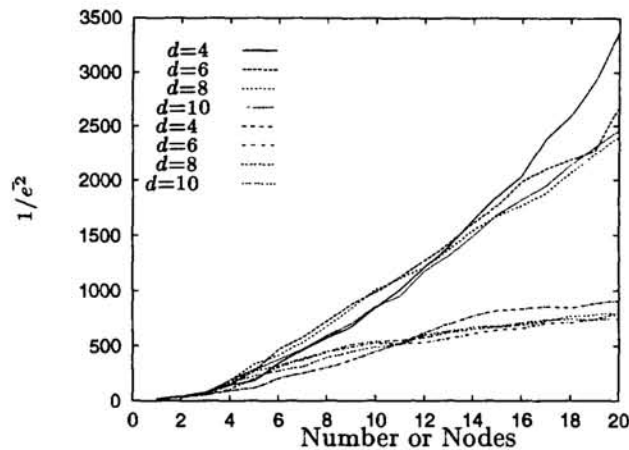

Figure 4: Upper (lower) curves are for training (test) data.

error, $\bar{e}^2$, was computed for both the training data and an independent set of test data of size $200d$. Plots of $1/\bar{e}^2$ versus the number of nodes are shown in Figure 4. The curves for the training data are clearly bounded below by a linear function of $n$ (as suggested by inverting the $O(1/n)$ result of Barron's). More importantly however, they show no significant dependence on the dimension $d$. The curves for the test data show the effect of the estimation error as they start to "bend over" around $n = 10$ nodes. Again however, they show no dependence on dimension.

## Acknowledgements

This work was inspired by the theoretical results of (Barron, 1993) for sigmoidal networks as well as the "Hinging Hyperplanes" work of (Breiman, 1993) , and the "Ramps" work of (Friedman and Breiman, 1994). This work was supported in part by ONR grant number N00014-95-1-1315.

## References

Barron, A.R. (1993) Universal approximation bounds for superpositions of a sigmoidal function. *IEEE Transactions on Information Theory* **39**(3):930–945.

Bellare, M. & Rogaway, P. (1993) The complexity of approximating a nonlinear program. In P.M. Pardalos (ed.), *Complexity in numerical optimization*, pp. 16–32, World Scientific Pub. Co.

Breiman, L. (1993) Hinging hyperplanes for regression, classification and function approximation. *IEEE Transactions on Information Theory* **39**(3):999–1013.

Breiman, L. & Friedman, J.H. (1994) Function approximation using RAMPS. *Snowbird Workshop on Machines that Learn.*

Edelsbrunner, H. (1987) In EATCS Monographs on Theoretical Computer Science V. 10, *Algorithms in Combinatorial Geometry.* Springer–Verlag.

Jones, L.K. (1992) A simple lemma on greedy approximation in Hilbert space and convergence rates for projection pursuit regression and neural network training. *The Annals of Statistics*, **20**:608–613.

Luenberger, D.G. (1984) *Introduction to Linear and Nonlinear Programming.* Addison–Wesley.